# A Reinforcement Learning Algorithm in Partially Observable Environments Using Short-Term Memory

**Nobuo Suematsu and Akira Hayashi**
Faculty of Computer Sciences
Hiroshima City University
3-4-1 Ozuka-higashi, Asaminami-ku, Hiroshima 731-3194 Japan
{suematsu,akira}@im.hiroshima-cu.ac.jp

## Abstract

We describe a Reinforcement Learning algorithm for partially observable environments using short-term memory, which we call BLHT. Since BLHT learns a stochastic model based on Bayesian Learning, the overfitting problem is reasonably solved. Moreover, BLHT has an efficient implementation. This paper shows that the model learned by BLHT converges to one which provides the most accurate predictions of percepts and rewards, given short-term memory.

## 1 INTRODUCTION

Research on Reinforcement Learning (RL) problem for partially observable environments is gaining more attention recently. This is mainly because the assumption that perfect and complete perception of the state of the environment is available for the learning agent, which many previous RL algorithms require, is not valid for many realistic environments.

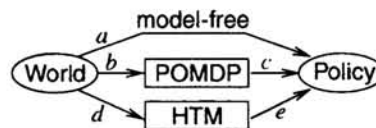

Figure 1: Three approaches

One of the approaches to the problem is the model-free approach (Singh et al. 1995; Jaakkola et al. 1995) (arrow $a$ in the Fig.1) which gives up state estimation and uses *memory-less* policies. We can not expect the approach to find a really effective policy when it is necessary to accumulate information to estimate the state. Model based approaches are superior in these environments.

A popular model based approach is via a Partially Observable Markov Decision Process (POMDP) model which represents the decision process of the agent. In Fig.1 the approach is described by the route from "World" to "Policy" through "POMDP". The approach has two serious difficulties. One is in the learning of POMDPs (arrow $b$ in Fig.1). Abe and

Warmuth (1992) shows that learning of probabilistic automata is NP-hard, which means that learning of POMDPs is also NP-hard. The other difficulty is in finding the optimal policy of a given POMDP model (arrow $c$ in Fig.1). Its PSAPCE-hardness is shown in Papadimitriou and Tsitsiklis (1987). Accordingly, the methods based on this approach (Chrisman 1992; McCallum 1993), will not scale well to large problems.

The approach using short-term memory is computationally more tractable. Of course we can construct environments in which long-term memory is essential. However, in many environments, because of their stochasticity, the significance of the past information decreases exponentially fast as the time goes. In such environments, memories of moderate length will work fine.

McCallum (1995) proposes "utile suffix memory" (USM) algorithm. USM uses a tree structure to represent short-term memories with variable length. USM's model learning is based on a statistical test, which requires time and space proportional to the learning steps. This makes it difficult to adapt USM to the environments which require long learning steps. USM suffers from the overfitting problem which is a difficult problem faced by most of model based learning methods. USM may overfit or underfit up to the significance level used for the statistical test and we can not know its proper level in advance.

In this paper, we introduce an algorithm called BLHT (Suematsu et al. 1997), in which the environment is modeled as a *history tree model* (HTM), a stochastic model with variable memory length. Although BLHT shares the tree structured representation of short-term memory with USM, the computational time required by BLHT is constant in each step and BLHT copes with environments which require large learning steps. In addition, because BLHT is based on Bayesian Learning, the overfitting problem is solved reasonably in it. A similar version of HTMs was introduced and has been used for learning of Hidden Markov Models in Ron et al. (1994). In their learning method, a tree is grown in a similar way with USM. If we try to adapt it to our RL problem, it will face the same problems with USM.

This paper shows that the HTM learned by BLHT converges to the optimal one in the sense that it provides the most accurate predictions of percepts and rewards, given short-term memory. BLHT can learn a HTM in an efficient way (arrow $d$ in Fig.1). And since HTMs compose a subset of Markov Decision Processes (MDPs), it can be efficiently solved by Dynamic Programming (DP) techniques (arrow $e$ in Fig.1). So, we can see BLHT as an approach to follow an easy way from "World" to "Policy" which goes around "POMDP".

## 2 THE POMDP MODEL

The decision process of an agent in a partially observable environment can be formulated as a POMDP. Let the finite set of states of the environment be $\mathcal{S}$, the finite set of agent's actions be $\mathcal{A}$, and the finite set of all possible percepts be $\mathcal{I}$. Let us denote the probability of and the reward for making transition from state $s$ to $s'$ using action $a$ by $p_{s'|sa}$ and $w_{sas'}$ respectively. We also denote the probability of obtaining percept $i$ after a transition from $s$ to $s'$ using action $a$ by $o_{i|sas'}$. Then, a POMDP model is specified by $\langle \mathcal{S}, \mathcal{A}, \mathcal{I}, \mathcal{P}, \mathcal{O}, \mathcal{W}, \boldsymbol{x}_0 \rangle$, where $\mathcal{P} = \{p_{s'|sa} \mid s, s' \in \mathcal{S}, a \in \mathcal{A}\}$, $\mathcal{O} = \{o_{i|sas'} | s, s' \in \mathcal{S}, a \in \mathcal{A}, i \in \mathcal{I}\}$, $\mathcal{W} = \{w_{sas'} | s, s' \in \mathcal{S}, a \in \mathcal{A}\}$, and $\boldsymbol{x}_0 = (x_{s_1}^0, \ldots, x_{s_{|S|-1}}^0)$ is the probability distribution of the initial state.

We denote the history of actions and percepts of the agent till time $t$, $(\ldots, a_{t-2}, i_{t-1}, a_{t-1}, i_t)$ by $D_t$. If the POMDP model, $M = \langle \mathcal{S}, \mathcal{A}, \mathcal{I}, \mathcal{P}, \mathcal{O}, \mathcal{W}, \boldsymbol{x}_i \rangle$ is given, one can compute the belief state, $\boldsymbol{x}_t = (x_{s_1}^t, \ldots, x_{s_{|S|-1}}^t)$ from $D_t$, which is the state estimation at time $t$. We denote the mapping from histories to belief states defined by POMDP model $M$ by $\boldsymbol{X}_M(\cdot)$, that is, $\boldsymbol{x}_t = \boldsymbol{X}_M(D_t)$. The belief state $\boldsymbol{x}_t$ is the most precise state estimation and it is known to be the sufficient statistics for the optimal policy in POMDPs (Bertsekas 1987). It is also known that the stochastic process $\{\boldsymbol{x}_t, t \geq 0\}$ is an MDP in the continuous

space, $\mathcal{X} \equiv \{(x_1, dots, x_{|\mathcal{S}|-1}) \mid x_1, \ldots, x_{|\mathcal{S}|-1} \geq 0, \sum_{j=1}^{|\mathcal{S}|-1} x_j \leq 1\}.$

# 3   BAYESIAN LEARNING OF HISTORY TREE MODELS (BLHT)

In this section, we summarize our RL algorithm for partially observable environments, which we call BLHT (Suematsu et al. 1997).

## 3.1   HISTORY TREE MODELS

BLHT is Bayesian Learning on a hypothesis space which is composed of predictive models, which we call History Tree Models (HTMs). Given short-term memory, a HTM provides the probability disctribution of the next percept and the expected immediate reward for each action. A HTM is represented by a tree structure called a history tree and parameters given for each leaf of the tree.

A history tree $h$ associates history $D_t$ with a leaf as follows. Starting from the root of $h$, we check the most recent percept, $i_t$ and follow the appropriate branch and then we check the action $a_{t-1}$ and follow the appropriate branch. This procedure is repeated till we reach a leaf. We denote the reached leaf by $\lambda_h(D_t)$ and the set of leaves of $h$ by $\mathcal{L}_h$.

Each leaf $l \in \mathcal{L}_h$ has parameters $\theta_{i|la}$ and $\omega_{la}$. $\theta_{i|la}$ denotes the probability of observing $i$ at time $t+1$ when $\lambda_h(D_t) = l$ and the last action $a_t$ was $a$. $\omega_{la}$ denotes the expected immediate reward for performing $a$ when $\lambda_h(D_t) = l$. Let $\Theta_h = \{\theta_{i|la} \mid i \in \mathcal{I}, l \in \mathcal{L}_h, a \in \mathcal{A}\}$.

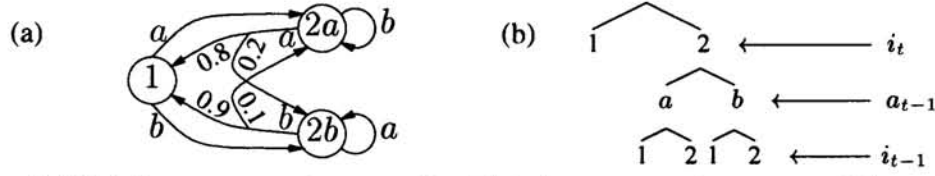

Figure 2: (a) A three-state environment, in which the agent receives percept 1 in state 1 and percept 2 in states $2a$ and $2b$. (b) A history tree which can represent the environment.

Fig. 2 shows a three-state environment (a) and a history tree which can represent the environment (b). We can construct a HTM which is equivalent with the environment by setting appropriate parameters in each leaf of the history tree.

## 3.2   BAYESIAN LEARNING

BLHT is designed as Bayesian Learning on the hypothesis space, $\mathcal{H}$, which is a set of history trees. First we show the posterior probability of a history tree $h \in \mathcal{H}$ given history $D_t$. To derive the posterior probability we set the prior density of $\Theta_h$ as

$$\rho(\Theta_h|h) = \prod_{l \in \mathcal{L}_h} \prod_{a \in \mathcal{A}} K_{la} \prod_{i \in \mathcal{I}} \theta_{i|la}^{\alpha_{i|la}-1},$$

where $K_{la}$ is the normalization constant and $\alpha_{i|la}$ is a hyper parameter to specify the prior density. Then we can have the posterior probability of $h$,

$$P(h|D_t, \mathcal{H}) = c_t P(h|\mathcal{H}) \prod_{l \in \mathcal{L}_h} \prod_{a \in \mathcal{A}} K_{la} \frac{\prod_{i \in \mathcal{I}} \Gamma(N_{i|la}^t + \alpha_{i|la})}{\Gamma(N_{la}^t + \alpha_{la})}, \tag{1}$$

where $c_t$ is the normalization constant, $\Gamma(\cdot)$ is the gamma function, $N_{i|la}^t$ is the number of times $i$ is observed after executing $a$ when $\lambda_h(D_{t'}) = l$ in the history $D_t$, $N_{la}^t = \sum_{i \in \mathcal{I}} N_{i|la}^t$, and $\alpha_{la} = \sum_{i \in \mathcal{I}} \alpha_{i|la}$.

Next, we show the estimates of the parameters. We use the average of $\theta_{i|la}$ with its posterior

density as the estimate, $\hat{\theta}^t_{i|la}$, which is expressed as

$$\hat{\theta}^t_{i|la} = \frac{N^t_{i|la} + \alpha_{i|la}}{N^t_{la} + \alpha_{la}}.$$

$\omega_{la}$ is estimated just by accumulating rewards received after executing $a$ when $\lambda_h(D_t) = l$, and dividing it by the number of times $a$ was performed when $\lambda_h(D_t) = l$, $N^t_{la}$. That is,

$$\hat{\omega}^t_{la} = \frac{1}{N^t_{la}} \sum_{k=1}^{N^t_{la}} r_{t_k+1},$$

where $t_k$ is the $k$-th occurrence of execution of $a$ when $\lambda_h(D_t) = l$.

### 3.3  LEARNING ALGORITHM

In principle, by evaluating Eq.(1) for all $h \in \mathcal{H}$, we can extract the MAP model. However, it is often impractical, because a proper hypothesis space $\mathcal{H}$ is very large when the agent has little prior knowledge concerning the environment. Fortunately, we can design an efficient learning algorithm by assuming that the hypothesis space, $\mathcal{H}$, is the set of pruned trees of a large history tree $h_{\mathcal{H}}$ and the ratio of prior probabilities of a history tree $h$ and $h'$ obtained by pruning off subtree $\Delta h$ from $h$ is given by a known function $q(\Delta h)$[1].

We define function $g(h|D_t, \mathcal{H})$ by taking logarithm of the R.H.S. of Eq.(1) without the normalization constant, which can be rewritten as

$$g(h|D_t, \mathcal{H}) = \log P(h|\mathcal{H}) + \sum_{l \in \mathcal{L}_h} \Lambda^t_l, \qquad (2)$$

where

$$\Lambda^t_l = \sum_{a \in \mathcal{A}} \log \left[ K_{la} \frac{\prod_{i \in \mathcal{I}} \Gamma(N^t_{i|la} + \alpha_{i|la})}{\Gamma(N^t_{la} + \alpha_{la})} \right]. \qquad (3)$$

Then, we can extract the MAP model by finding the history tree which maximizes $g$. Eq.(2) shows that $g(h|D_t, \mathcal{H})$ can be evaluated by summing up $\Lambda^t_l$ over $\mathcal{L}_h$. Accordingly, we can implement an efficient algorithm using the tree $h_{\mathcal{H}}$ whose each (internal or leaf) node $l$ stores $\Lambda_l$, $N_{i|la}$, $\alpha_{i|la}$, and $\omega_{la}$.

Suppose that the agent observed $i_{t+1}$ when the last action was $a_t$. Then, from Eq.(3),

$$\Lambda^{t+1}_l = \begin{cases} \Lambda^t_l + \log \frac{N^t_{i_{t+1}|la_t} + \alpha_{i_{t+1}|la_t}}{N^t_{la_t} + \alpha_{la_t}} & \text{for } l \in \mathcal{N}_{D_t} \\ \Lambda^t_l & \text{otherwise} \end{cases}, \qquad (4)$$

where $\mathcal{N}_{D_t}$ is the set of nodes on the path from the root to leaf $\lambda_{h_{\mathcal{H}}}(D_t)$. Thus, $h_{\mathcal{H}}$ is updated just by evaluating Eq(4), adding 1 to $N_{i|la}$, and recalculating $\omega_{la}$ in nodes of $\mathcal{N}_{D_t}$.

After $h_{\mathcal{H}}$ is updated, we can extract the MAP model using the procedure "Find-MAP-Subtree" shown in Fig. 3(a). We show the learning algorithm in Fig.3(b), in which the MAP model is extracted and policy $\pi$ is updated only when a given condition is satisfied.

## 4  LIMIT THEOREMS

In this section, we describe limit theorems of BLHT. Throughout the section, we assume that policy $\pi$ is used while learning and the stochastic process $\{(s_t, a_t, i_{t+1}), t \geq 0\}$ is ergodic under $\pi$.

First we show a theorem which ensures that the history tree model learned by BLHT does not miss any relevant memories (see Suematsu et al. (1997) for the proof).

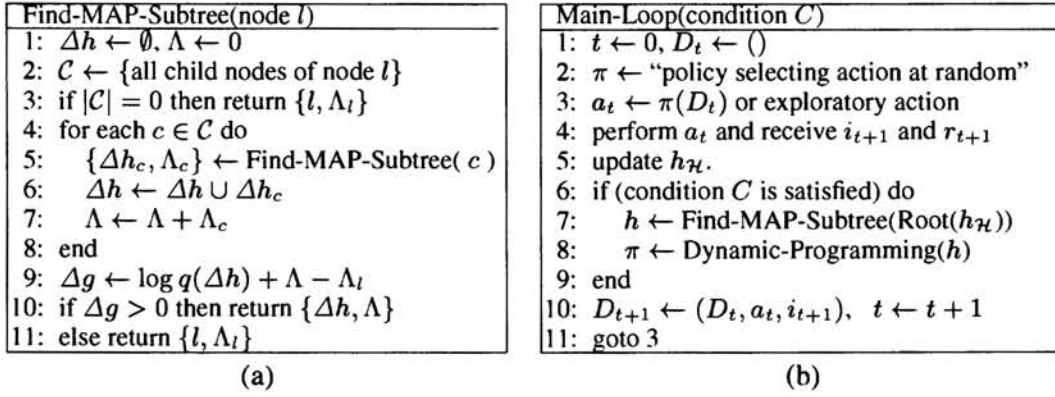

(a)  (b)

Figure 3: The procedure to find MAP subtree (a) and the main loop (b).

**Theorem 1** *For any* $h \in \mathcal{H}$,

$$\lim_{t \to \infty} \frac{1}{t} g(h|D_t, \mathcal{H}) = -H_h(I|L, A),$$

*where* $H_h(I|L, A)$ *is the conditional entropy of* $i_{t+1}$ *given* $l_t = \lambda_h(D_t)$ *and* $a_t$ *defined by*

$$H_h(I|L, A) \equiv E_\pi \left\{ \sum_{i \in \mathcal{I}} -P_\pi(i_{t+1} = i \mid l_t, a_t) \log P_\pi(i_{t+1} = i \mid l_t, a_t) \right\},$$

*where* $P_\pi(\cdot)$ *and* $E_\pi(\cdot)$ *denotes probability and expected value under* $\pi$ *respectively.*

Let the history tree shown in Fig.2(b) be $h^*$ and a history tree obtained by pruning a subtree of $h^*$ be $h^-$. Then, for the environment shown in Fig.2(a) $H_{h^-}(I|L, A) > H_{h^*}(I|L, A)$, because $h^-$ misses some relevant memories and it makes the conditional entropy increase. Since BLHT learns the history tree which maximizes $g(h|D_t, \mathcal{H})$ (minimizes $H_h(I|L, A)$), the learned history tree does not miss any relevant memory.

Next we show a limit theorem concerning the estimates of the parameters. We denote the true POMDP model by $M = \langle \mathcal{S}, \mathcal{A}, \mathcal{I}, \mathcal{P}, \mathcal{O}, \mathcal{W}, x_i \rangle$ and define the following parameters,

$$\sigma_{i|sa} \equiv P(i_{t+1} = i \mid s_t = s, a_t = a) = \sum_{s' \in \mathcal{S}} p_{s'|sa} o_{i|sas'}$$

$$\mu_{sa} \equiv E(r_{t+1}|s_t = s, a_t = a) = \sum_{s' \in \mathcal{S}} w_{sas'} p_{s'|sa}.$$

Then, the following theorem holds.

**Theorem 2** *For any leaf* $l \in \mathcal{L}_h$, $a \in \mathcal{A}$, $i \in \mathcal{I}$

$$\lim_{t \to \infty} \hat{\theta}^t_{i|la} = \sum_{s \in \mathcal{S}} \sigma_{i|sa} y^\pi_{s|la}, \tag{5}$$

$$\lim_{t \to \infty} \hat{\omega}^t_{la} = \sum_{s \in \mathcal{S}} \mu_{sa} y^\pi_{s|la}, \tag{6}$$

*where* $y^\pi_{s|la} \equiv P_\pi(s_t = s|\lambda_h(D_t) = l, a_t = a)$.

**Outline of proof :** Using the Ergodic Theorem, We have

$$\lim_{t \to \infty} \hat{\theta}^t_{i|la} = P_\pi(i_{t+1} = i|\lambda_h(D_t) = l, a_t = a).$$

By expanding R.H.S of the above equation using the chain rule, we can derive Eq.(5). Eq.(6) can be derived in a similar way.                                                               ■

To explain what Theorem 2 means clearly, we show the relationship between $y_{s|la}^{\pi}$ and the belief state $x_t$.

$$P_{\pi}(s_t = s | \lambda_h(D_t) = l, a_t = a, x_0 = x_i)$$

$$= \sum_{D \in \mathcal{D}_l^t} P(s_t = s | D_t = D, a_t = a, x_0 = x_i) P_{\pi}(D_t = D | l_t = l, a_t = a, x_0 = x_i)$$

$$= \int_{\mathcal{X}} \sum_{D \in \mathcal{D}_l^t} \mathbb{1}_{\mathcal{D}_x^t}(D) \{ X_M(D) \}_s P_{\pi}(D_t = D | l_t = l, a_t = a, x_0 = x_i) dx$$

$$= \int_{\mathcal{X}} x_s P_{\pi}(x_t = x | l_t = l, a_t = a, x_0 = x_i) dx,$$

where $\mathcal{D}_l^t \equiv \{ D_t | \lambda_h(D_t) = l \}$, $\mathbb{1}_B(\cdot)$ is the indicator function of a set $B$, $\mathcal{D}_x^t \equiv \{ D_t | X_M(D_t) = x \}$, and $dx = dx_1 \cdots dx_{|\mathcal{S}|-1}$. Under the ergodic assumption, by taking $\lim_{t \to \infty}$ of the above equation, we have

$$y_{la}^{\pi} = \int_{\mathcal{X}} x \, \Phi_{la}^{\pi}(x) dx \tag{7}$$

where $y_{la}^{\pi} = (y_{s_1|la}^{\pi}, \dots, y_{s_{|\mathcal{S}|-1}|la}^{\pi})$ and $\Phi_{la}^{\pi}(x) = P_{\pi}(x_t = x | \lambda_h(D_t) = l, a_t = a)$.

We see from Eq.(7) that $y_{la}^{\pi}$ is the average of belief state $x_t$ with conditional density $\Phi_{la}^{\pi}$, that is, the belief states distributed according to $\Phi_{la}^{\pi}$ are represented by $y_{la}^{\pi}$. When short-term memory of $l$ gives the dominant information of $D_t$, $\Phi_{la}^{\pi}$ is concentrated and $y_{la}^{\pi}$ is a reasonable approximation of the belief states. An extreme of the case is when $\Phi_{la}^{\pi}$ is non-zero only at a point in $\mathcal{X}$. Then $y_{la}^{\pi} = x_t$ when $\lambda_h(D_t) = l$.

Please note that given short-term memory represented by $l$ and $a$, $y_{la}^{\pi}$ is the most accurate state estimation. Consequently, Theorem 1 and 2 ensure that learned HTM converges to the model which provides the most accurate predictions of percepts and rewards among $\mathcal{H}$. This fact provides a solid basis for BLHT, and we believe BLHT can be compared favorably with other methods using short-term memory. Of course, Theorem 1 and 2 also say that BLHT will find the optimal policy if the environment is Markovian or semi-Markovian whose order is small enough for the equivalent model to be contained in $\mathcal{H}$.

## 5  EXPERIMENT

We made experiments in various environments. In this paper, we show one of them to demonstrate the effectiveness of BLHT. The environment we used is the grid world shown in Fig.4(a). The agent has four actions to change its location to one of the four neighboring grids, which will fail with probability 0.2. On failure, the agent does not change the location with probability 0.1 or goes to one of the two grids which are perpendicular to the direction the agent is trying to go with probability 0.1. The agent can detect merely the existence of the four surrounding walls. The agent receives a reward of 10 when he reaches the goal which is the grid marked with "G" and - 1 when he tries to go to a grid occupied by an obstacle. At the goal, any action will relocate the agent to one of the starting states which are marked with "S" at random. In order to achieve high performance in the environment, the agent has to select different actions for an identical immediate percept, because many of the states are aliased (i.e. they look identical by the immediate percepts). The environment has 50 states, which is among the largest problems shown in the literature of the model based RL techniques for partially observable environments.

Fig.4(b) shows the learning curve which is obtained by averaging over 10 independent runs. While learning, the agent updated the policy every 10 trials (10 visits to the goal) and the

policy was evaluated through a run of 100,000 steps. Actions were selected using the policy or at random and the probability of selecting at random was decreased exponentially as the time goes. We used the tree which has homogeneous depth of 5 as $h_{\mathcal{H}}$. In Fig.4(b), the horizontal broken line indicates the average reward for the MDP model obtained by assuming perfect and complete perception. It gives an upper bound for the original problem, and it will be higher than the optimal one for the original problem. The learning curve shown there is close to the upper bound in the later stage.

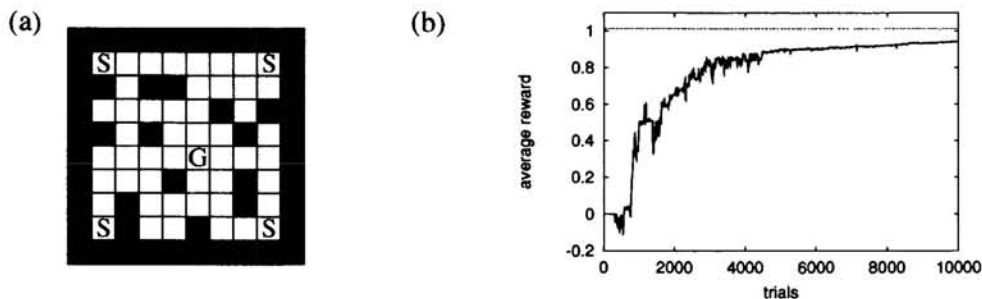

Figure 4: The grid world (a) and the learning curve (b).

## 6 SUMMARY

This paper has described a RL algorithm for partially observable environments using short-term memory, which we call BLHT. We have proved that the model learned by BLHT converges to the optimal model in given hypothesis space, $\mathcal{H}$, which provides the most accurate predictions of percepts and rewards, given short-term memory. We believe this fact provides a solid basis for BLHT, and BLHT can be compared favorably with other methods using short-term memory.

## Footnotes

[1]The condition is satisfied, for example, when $P(h|\mathcal{H}) \propto \gamma^{|h|}$ where $0 < \gamma \leq 1$ and $|h|$ denotes the size of $h$.

## References

Abe, N. and M. K. Warmuth (1992). On the computational compleixy of apporximating distributions by probabilistic automata. *Machine Learning*, 9:205–260.

Bertsekas, D. P. (1987). *Dyanamic Programming*. Prentice-Hall.

Chrisman, L. (1992). Reinforcemnt learning with perceptual aliasing: The perceptual distinctions approach. In *Proc. the 10th National Conference on Artificial Intelligence*.

Jaakkola, T., S. P. Singh, and M. I. Jordan (1995). Reinforcement learning algorithm for parially observable markov decision problems. In *Advances in Neural Information Processing Systems 7*, pp. 345–352.

McCallum, R. A. (1993). Overcoming incomplete perception with utile distiction memory. In *Proc. the 10th International Conference on Machine Learning*.

McCallum, R. A. (1995). Instance-based utile distinctions for reinforcement learning with hidden state. In *Proc. the 12th International Conference on Machine Learning*.

Papadimitriou, C. H. and J. N. Tsitsiklis (1987). The complexity of markov decision processes. *Mathematics of Operations Research*, 12(3):441–450.

Ron, D., Y. Singer, and N. Tishby (1994). Learning probabilistic automata with variable memory length. In *Proc. of Computational Learning Theory*, pp. 35–46.

Singh, S. P., T. Jaakkola, and M. I. Jordan (1995). Learning without state-estimation in partially observable markov decision processes. In *Proc. the 12th International Conference on Machine Learning*, pp. 284–292.

Suematsu, N., A. Hayashi, and S. Li (1997). A Bayesian approch to model learning in non-markovian environments. In *Proc. the 14th International Conference on Machine Learning*, pp. 349–357.
